# Time-rescaling methods for the estimation and assessment of non-Poisson neural encoding models

**Jonathan W. Pillow**
Departments of Psychology and Neurobiology
University of Texas at Austin
`pillow@mail.utexas.edu`

## Abstract

Recent work on the statistical modeling of neural responses has focused on modulated renewal processes in which the spike rate is a function of the stimulus and recent spiking history. Typically, these models incorporate spike-history dependencies via either: (A) a conditionally-Poisson process with rate dependent on a linear projection of the spike train history (e.g., generalized linear model); or (B) a modulated non-Poisson renewal process (e.g., inhomogeneous gamma process). Here we show that the two approaches can be combined, resulting in a *conditional renewal* (CR) model for neural spike trains. This model captures both real-time and rescaled-time history effects, and can be fit by maximum likelihood using a simple application of the time-rescaling theorem [1]. We show that for any modulated renewal process model, the log-likelihood is concave in the linear filter parameters only under certain restrictive conditions on the renewal density (ruling out many popular choices, e.g. gamma with shape $\kappa \neq 1$), suggesting that real-time history effects are easier to estimate than non-Poisson renewal properties. Moreover, we show that goodness-of-fit tests based on the time-rescaling theorem [1] quantify relative-time effects, but do not reliably assess accuracy in spike prediction or stimulus-response modeling. We illustrate the CR model with applications to both real and simulated neural data.

## 1   Introduction

A central problem in computational neuroscience is to develop functional models that can accurately describe the relationship between external variables and neural spike trains. All attempts to measure information transmission in the nervous system are fundamentally attempts to quantify this relationship, which can be expressed by the conditional probability $P(\{t_i\}|X)$, where $\{t_i\}$ is a set of spike times generated in response to an external stimulus $X$.

Recent work on the neural coding problem has focused on extensions of the Linear-Nonlinear-Poisson (LNP) "cascade" encoding model, which describes the neural encoding process using a linear receptive field, a point nonlinearity, and an inhomogeneous Poisson spiking process [2, 3]. While this model provides a simple, tractable tool for characterizing neural responses, one obvious shortcoming is the assumption of Poisson spiking. Neural spike trains exhibit spike-history dependencies (e.g., refractoriness, bursting, adaptation), violating the Poisson assumption that spikes in disjoint time intervals are independent. Such dependencies, moreover, have been shown to be essential for extracting complete stimulus information from spike trains in a variety of brain areas [4, 5, 6, 7, 8, 9, 10, 11].

Previous work has considered two basic approaches for incorporating spike-history dependencies into neural encoding models. One approach is to model spiking as a non-Poisson inhomogeneous renewal process (e.g., a modulated gamma process [12, 13, 14, 15]). Under this approach, spike

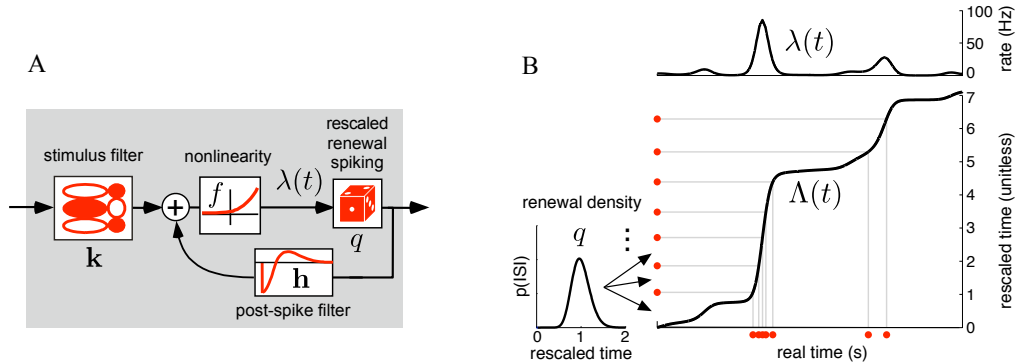

**Figure 1:** The conditional renewal (CR) model and time-rescaling transform. (**A**) Stimuli are convolved with a filter **k** then passed through a nonlinearity $f$, whose output is the rate $\lambda(t)$ for an inhomogeneous spiking process with renewal density $q$. The post-spike filter **h** provides recurrent additive input to $f$ for every spike emitted. (**B**) Illustration of the time-rescaling transform and its inverse. Top: the intensity $\lambda(t)$ (here independent of spike history) in response to a one-second stimulus. Bottom left: interspike intervals (left, intervals between red dots) are drawn *i.i.d.* in rescaled time from renewal density $q$, here set to gamma with shape $\kappa = 20$. Samples are mapped to spikes in real time (bottom) via $\Lambda^{-1}(t)$, the inverse of the cumulative intensity. Alternatively, $\Lambda(t)$ maps the true spike times (bottom) to samples from a homogeneous renewal process in rescaled time (left edge).

times are Markovian, depending on the most recent spike time via a (non-exponential) renewal density, which may be rescaled in proportion to the instantaneous spike rate. A second approach is to use a conditionally Poisson process in which the intensity (or spike rate) is a function of the recent spiking history [4, 16, 17, 18, 19, 20]. The output of such a model is a *conditionally Poisson* process, but not Poisson, since the spike rate itself depends on the spike history.

The time-rescaling theorem, described elegantly for applications to neuroscience in [1], provides a powerful tool for connecting these two basic approaches, which is the primary focus of this paper. We begin by reviewing inhomogeneous renewal models and generalized linear model point process models for neural spike trains.

## 2   Point process neural encoding models

### 2.1   Definitions and Terminology

Let $\{t_i\}$ be a sequence of spike times on the interval $(0, T]$, with $0 < t_0 < t_1 < \ldots, < t_n \leq T$, and let $\lambda(t)$ denote the *intensity* (or "spike rate") for the point process, where $\lambda(t) \geq 0, \forall t$. Generally, this intensity is a function of some external variable (e.g., a visual stimulus). The *cumulative intensity* function is given by the integrated intensity,

$$\Lambda(t) = \int_0^t \lambda(s)ds, \tag{1}$$

and is also known as the *time-rescaling transform* [1]. This function rescales the original spike times into spikes from a (homogeneous) renewal process, that is, a process in which the intervals are *i.i.d.* samples from a fixed distribution. Let $\{u_i\}$ denote the inter-spike intervals (ISIs) of the rescaled process, which are given by the integral of the intensity between successive spikes, i.e.,

$$u_i = \Lambda_{t_{i-1}}(t_i) = \int_{t_{i-1}}^{t_i} \lambda(s)ds. \tag{2}$$

Intuitively, this transformation stretches time in proportion to the spike rate $\lambda(t)$, so that when the rate $\lambda(t)$ is high, ISIs are lengthened and when $\lambda(t)$ is low, ISIs are compressed. (See fig. 1B for illustration).

Let $q(u)$ denote the *renewal density*, the probability density function from which the rescaled-time intervals $\{u_i\}$ are drawn. A Poisson process arises if $q$ is exponential, $q(u) = e^{-u}$; for any other density, the probability of spiking depends on the most recent spike time. For example, if $q(u)$ is zero for $u \in [0, a]$, the neuron exhibits a refractory period (whose duration varies with $\lambda(t)$).

To sample from this model (illustrated in fig. 1B), we can draw independent intervals $u_i$ from renewal density $q(u)$, then apply the inverse time-rescaling transform to obtain ISIs in real time:

$$(t_i - t_{i-1}) = \Lambda_{t_{i-1}}^{-1}(u_i), \tag{3}$$

where $\Lambda_{t_{i-1}}^{-1}(t)$ is the inverse of time-rescaling transform (eq 2).[1]

We will generally define the intensity function (which we will refer to as the *base intensity*[2]) in terms of a linear-nonlinear cascade, with linear dependence on some external covariates of the response (optionally including spike-history), followed by a point nonlinearity. The intensity in this case can be written:

$$\lambda(t) = f(\mathbf{x}_t \cdot \mathbf{k} + \mathbf{y}_t \cdot \mathbf{h}), \tag{4}$$

where $\mathbf{x}_t$ is a vector representing the stimulus at time $t$, $\mathbf{k}$ is a stimulus filter, $\mathbf{y}_t$ is a vector representing the spike history at $t$, and $\mathbf{h}$ is a spike-history filter. We assume that the nonlinearity $f$ is fixed.

## 2.2 The conditional renewal model

We refer to the most general version of this model, in which $\lambda(t)$ is allowed to depend on both the stimulus and spike train history, and $q(u)$ is an arbitrary (finite-mean) density on $\mathbb{R}^+$, as a *conditional renewal* (CR) model (see fig. 1A). The output of this model forms an inhomogeneous renewal process conditioned on the process history. Although it is mathematically straightforward to define such a model, to our knowledge, no previous work has sought to incorporate both real-time (via $\mathbf{h}$) and rescaled-time (via $q$) dependencies in a single model.

Specific (restricted) cases of the CR model include the generalized linear model (GLM) [17], and the modulated renewal model with $\lambda = f(\mathbf{x} \cdot \mathbf{k})$ and $q$ a right-skewed, non-exponential renewal density [13, 15]. (Popular choices for $q$ include gamma, inverse Gaussian, and log-normal distributions).

The conditional probability distribution over spike times $\{t_i\}$ given the external variables $X$ can be derived using the time-rescaling transformation. In rescaled time, the CR model specifies a probability over the ISIs,

$$P(\{u_i\}|X) = \prod_{i=1}^{n} q(u_i). \tag{5}$$

A change-of-variables $t_i = \Lambda_{t_{i-1}}^{-1}(u_i) + t_{i-1}$ (eq. 3) provides the conditional probability over spike times:

$$P(\{t_i\}|X) = \prod_{i=1}^{n} \lambda(t_i) q(\Lambda_{t_{i-1}}(t_i)). \tag{6}$$

This probability, considered as a function of the parameters defining $\lambda(t)$ and $q(u)$, is the likelihood function for the CR model, as derived in [13].[3] The log-likelihood function can be approximated in discrete time, with bin-size $dt$ taken small enough to ensure $\leq 1$ spike per bin:

$$\log P(\{t_i\}|X) = \sum_{i=1}^{n} \log \lambda(t_i) + \sum_{i=1}^{n} \log q \left( \sum_{j=t_{i-1}+1}^{t_i} \lambda(j) dt \right), \tag{7}$$

where $t_i$ indicates the bin for the $i$th spike. This approximation becomes exact in the limit as $dt \to 0$.

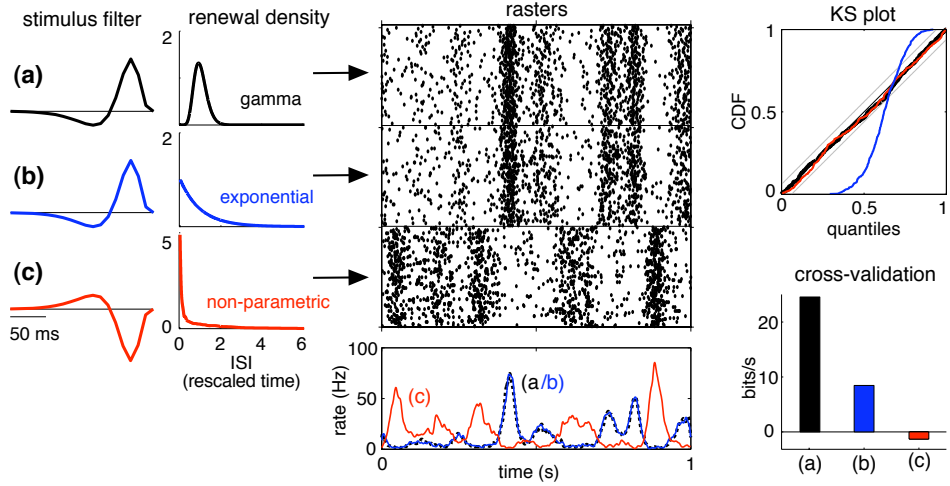

**Figure 2:** Time-rescaling and likelihood-based goodness-of-fit tests with simulated data. **: Left:** Stimulus filter and renewal density for three point process models (all with nonlinearity $f(x) = e^x$ and history-independent intensity). "True" spikes were generated from **(a)**, a conditional renewal model with a gamma renewal density ($\kappa = 10$). These responses were fit by: **(b)**, a Poisson model with the correct stimulus filter; and **(c)**, a modulated renewal process with *incorrect* stimulus filter (set to the negative of the correct filter), and renewal density estimated nonparametrically from the transformed intervals (eq. 10). **Middle:** Repeated responses from all three models to a novel 1-s stimulus, showing that spike rate is well predicted by (b) but not by (c). **Right**: KS plots (above) show time-rescaling based goodness-of-fit. Here, (b) fails badly, while (c) passes easily, with cdf entirely within within 99% confidence region (gray lines). Likelihood-based cross-validation tests (below) show that (b) preserves roughly 1/3 as much information about spike times as (a), while (c) carries slightly less information than a homogeneous Poisson process with the correct spike rate.

## 3  Convexity condition for inhomogeneous renewal models

We now turn to the tractability of estimating the CR model parameters from data. Here, we present an extension to the results of [21], which proved a convexity condition for maximum-likelihood estimation of a conditionally Poisson encoding model (i.e., generalized linear model). Specifically, [21] showed that the log-likelihood for the filter parameters $\theta = \{\mathbf{k}, \mathbf{h}\}$ is concave (i.e., has no non-global local maxima) if the nonlinear function $f$ is both convex and log-concave (meaning $\log f$ is concave). Under these conditions[4], minimizing the negative log-likelihood is a convex optimization problem.

By extension, we can ask whether the estimation problem remains convex when we relax the Poisson assumption and allow for a non-exponential renewal density $q$. Let us write the log-likelihood function for the linear filter parameters $\theta = [\mathbf{k}^T, \mathbf{h}^T]^T$ as

$$L_{\{D,q\}}(\theta) = \sum_i^n \log f(X(t_i) \cdot \theta) + \sum_{i=1}^n \log q \left( \int_{t_{i-1}}^{t_i} f(X(t) \cdot \theta)dt \right), \qquad (8)$$

where $X(t) = [\mathbf{x}_t^T, \mathbf{y}_t^T]^T$ is a vector containing the relevant stimulus and spike history at time $t$, and $D = \{\{t_i\}, \{X(t)\}\}$ represents the full set of observed data. The condition we obtain is:

**Theorem 1.** *The CR model log-likelihood $L_{\{D,q\}}(\theta)$ is concave in the filter parameters $\theta$, for any observed data $D$, if: (1) the nonlinearity $f$ is convex and log-concave; and (2) the renewal density $q$ is log-concave and non-increasing on $(0, \infty]$.*

*Proof.* It suffices to show that both terms in the equation (8) are concave in $\theta$, since the sum of two concave functions is concave. The first term is obviously concave, since $\log f$ is concave. For the

second term, note that $\int f(X \cdot \theta)$ is a convex function, since it is the integral of a convex function over a convex region. Then $\log q[\int f(X \cdot \theta)]$ is a concave, non-increasing function of a convex function, since $\log q$ is concave and non-increasing; such a function is necessarily concave.[5] The second term is therefore also a sum of concave functions, and thus concave. $\square$

Maximum likelihood filter estimation under the CR model is therefore a convex problem so long as the renewal density $q$ is both log-concave and non-increasing. This restriction rules out a variety of renewal densities that are commonly employed to model neural data [13, 14, 15]. Specifically, the log-normal and inverse-Gaussian densities both have increasing regimes on a subset of $[0, \infty)$, as does the gamma density $q(u) \propto u^{\kappa-1} e^{-u\kappa}$ when $\kappa > 1$. For $\kappa < 1$, gamma fails to be log-concave, meaning that the only gamma density satisfying both conditions is the exponential ($\kappa = 1$).

There are nevertheless many densities (besides the exponential) for which these conditions are met, including

- $q(u) \propto e^{-u^p/\sigma^2}$, for any $p \geq 1$
- $q(u) =$ uniform density
- $q(u) \propto \lfloor f(u) \rfloor$, or $q(u) \propto e^{f(u)}$, for any concave, decreasing function $f(u)$

Unfortunately, no density in this family can exhibit refractory effects, since this would require a $q$ that is initially zero and then rises. From an estimation standpoint, this suggests that it is easier to incorporate certain well-known spike-history dependencies using recurrent spike-history filters (i.e., using the GLM framework) than via a non-Poisson renewal density.

An important corollary of this convexity result is that the *decoding* problem of estimating stimuli $\{x_t\}$ from a set of observed spike times $\{t_i\}$ using the maximum of the posterior (i.e., computing the MAP estimate) is also a convex problem under the same restrictions on $f$ and $q$, so long as the prior over stimuli is log-concave.

## 4   Nonparametric Estimation of the CR model

In practice, we may wish to optimize both the filter parameters governing the base intensity $\lambda(t)$ and the renewal density $q$, which is not in general a convex problem. We may proceed, however, bearing in mind that gradient ascent may not achieve the global maximum of the likelihood function.

Here we formulate a slightly different interval-rescaling function that allows us to non-parametrically estimate renewal properties using a density on the unit interval. Let us define the mapping

$$v_i = 1 - \exp(-\Lambda_{t_{i-1}}(t_i)), \tag{9}$$

which is the cumulative density function (cdf) for the intervals from a conditionally Poisson process with cumulative intensity $\Lambda(t)$. This function maps spikes from a conditionally Poisson process to *i.i.d.* samples from $U[0, 1]$. Any discrepancy between the distribution of $\{v_i\}$ and the uniform distribution represents failures of a Poisson model to correctly describe the renewal statistics. (This is the central idea underlying time-rescaling based goodness-of-fit test, which we will discuss shortly).

We propose to estimate a density $\phi(v)$ for the rescaled intervals $\{v_i\}$ using cubic splines (piecewise 3rd-order polynomials with continuous 2nd derivatives), with evenly spaced knots on the interval $[0, 1]$.[6] This allows us to rewrite the likelihood function (6) as the product of two identifiable terms:

$$P(\{t_i\}|X) = \left( \prod_{i=1}^{n} \lambda(t_i) \, e^{-\Lambda_0(T)} \right) \left( \prod_{i=1}^{n} \phi(v_i) \right), \tag{10}$$

where the first term is the likelihood under the conditional Poisson model [17], and the second is the probability of the rescaled intervals $\{v_i\}$ under the density $\phi(v)$. This formulation allows us to separate the (real-time) contributions of the intensity function under the assumption of conditionally

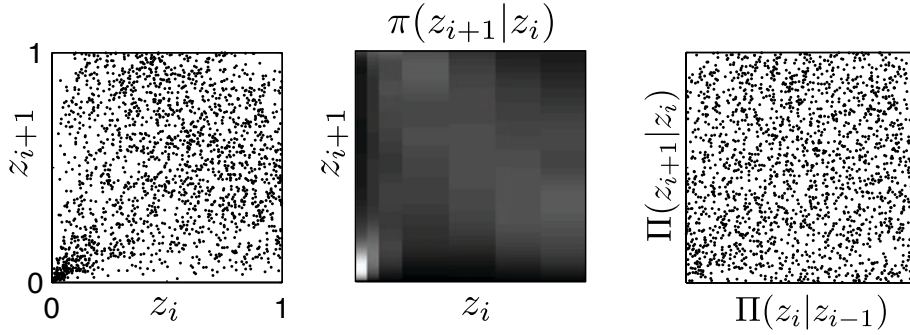

**Figure 3: Left**: pairwise dependencies between successive rescaled ISIs from model ("a", see fig. 2) when fit by a non-Poisson renewal model "c". **Center**: fitted model of the conditional distribution over rescaled ISIs given the previous ISI, discretized into 7 intervals for the previous ISI. **Right**: rescaling the intervals using the cdf $\Pi$, obtained from the conditional $\pi(z_{i+1}|z_i)$, produces successive ISIs which are much more independent. This transformation adds roughly 3 bits/s to the likelihood-based cross-validation performance of model (c).

Poisson spiking, from the (rescaled-time) contributions of a non-Poisson renewal density. (For a conditionally Poisson process, $\pi$ is the uniform density on $[0\ 1]$, and makes zero contribution to the total log-likelihood).

We fit this model to simulated data (fig. 2), and to real neural data using alternating coordinate ascent of the filter parameters and the renewal density parameters (fig. 4). In fig. 2, we plot the renewal distribution $q(u)$ (red trace), which can be obtained from the estimated $\gamma(v)$ via the transformation $q(u) = \gamma(1\ e^{-u})e^{-u}$.

## 4.1 Incorporating dependencies between intervals

The cdf defined by the CR model, $\Gamma(v) = \int_0^v \gamma(s)ds$, maps the transformed ISIs $v_i$ so that the marginal distribution over $z_i = \Gamma(v_i)$ is uniform on $[0\ 1]$. However, there is no guarantee that the resulting random variables are independent, as assumed in the likelihood (eq. 10). We can examine dependencies between successive ISIs by making a scatter plot of pairs $(z_i\ z_{i+1})$ (see fig. 3). Departures from independence can then be modeled by introducing a nonparametric estimator for the conditional distribution $\phi(z_i z_{i-1})$. In this case, the likelihood becomes

$$P(\ t_i\ X) = \prod_{i=1}^{n} \lambda(t_i)\ e^{-\Lambda_0(T)} \prod_{i=1}^{n} \gamma(v_i) \prod_{i=2}^{n} \phi(z_i z_{i-1}) \qquad (11)$$

which now has three terms, corresponding (respectively) to the effects of the base intensity, non-conditionally Poisson renewal properties, and dependencies between successive intervals.

## 5 The time-rescaling goodness-of-fit test

If a particular point-process model provides an accurate description of a neuron's response, then the cumulative intensity function defines a mapping from the real time to rescaled-time such that the rescaled interspike intervals have a common distribution. Time-rescaling can therefore be used as a tool for assessing the goodness-of-fit of a point process model [1, 22]. Specifically, after remapping a set of observed spike times according to the (model-defined) cumulative intensity, one can perform a distributional test (e.g., Kolmogorov-Smirnov, or KS test) to assess whether the rescaled intervals have the expected distribution[7]. For example, for a conditionally Poisson model, the KS test can be applied to the rescaled intervals $v_i$ (eq. 9) to assess their fit to a uniform distribution.

This approach to model validation has grown in popularity in recent years [14, 23], and has in some instances been used as the only metric for comparing models. We wish to point out that time-rescaling based tests are sensitive to one kind of error (i.e., errors in modeling rescaled ISIs), but may be insensitive to other kinds of model error (i.e., errors in modeling the stimulus-dependent spike rate). Inspection of the CR model likelihood (eq. 10), makes it clear that time-rescaling based goodness-of-fit tests are sensitive only to accuracy with which $\phi(v)$ (or equivalently, $q(u)$) models the rescaled intervals. The test can in fact be independent of the accuracy with which the model describes the transformation from stimulus to spikes, a point that we illustrate with an (admittedly contrived) example in fig. 2.

For this example, spikes were genereated from a "true" model (denoted "a"), a CR model with a biphasic stimulus filter and a gamma renewal density ($\kappa = 10$). Responses from this model were fit by two sub-optimal approximate models: "b", a Poisson (LNP) model, which was specified to have the correct stimulus filter; and "c", a CR model in which the stimulus filter was mis-specified (set to the negative of the true filter), and a renewal density $\phi(v)$ was estimated non-parametrically from the rescaled intervals $\{v_i\}$ (rescaled under the intensity defined by this model).

Although the time-varying spike-rate predictions of model (c) were badly mis-matched to those of model (a) (fig. 2, middle), a KS-plot (upper right) shows that (c) exhibits near perfect goodness-of-fit on a time-rescaling test, which the Poisson model (b) fails badly. We cross-validated these models by computing the log-likelihood of novel data, which provides a measure of predictive information about novel spike trains in units of bits/s [24, 18]. Using this measure, the "true" model (a) provides approximately 24 bits/s about the spike response to a novel stimulus. The Poisson model (b) captures only 8 bits/s, but is still much more accurate than the mis-specified renewal model (c), for which the information is slightly negative (indicating that performance is slightly worse than that of a homogeneous Poisson process with the correct rate).

Fig. 3 shows that model (c) can be improved by modeling the dependencies between successive rescaled interspike intervals. We constructed a spline-based non-parametric estimate of the density $\pi(z_{i+1}|z_i)$, where $z_i = \Phi(v_i)$. (We discretized $z_i$ into 7 bins, based on visual inspection of the pair-wise dependency structure, and fit a cubic spline with 10 evenly spaced knots on [0,1] to the density within each bin). Rescaling these intervals using the cdf of the augmented model yields intervals that are both uniform on $[0, 1]$ and approximately independent (fig. 3, right; independence for non-successive intervals not shown). The augmented model raises the cross-validation score of model (c) to 1 bit/s, meaning that by incorporating dependencies between intervals, the model carries slightly more predictive information than a homogeneous Poisson model, despite the mis-specified stimulus filter. However, this model—despite passing time-rescaling tests of both marginal distribution and independence—still carries less information about spike times than the inhomogeneous Poisson model (b).

## 6 Application to neural data

Figure 4 shows several specific cases of the CR model fit to spiking data from an ON parasol cell in primate retina, which was visually stimulated with binary spatio-temporal white noise (i.e., flicker-ing checkerboard, [18]). We fit parameters for the CR model with and without spike-history filters, and with and without a non-Poisson renewal density (estimated non-parametrically as described above).

As expected, a non-parametric renewal density allows for remapping of ISIs to the correct (uniform) marginal distribution in rescaled time (fig. 4, left), and leads to near-perfect scores on the time-rescaling goodness-of-fit test (middle). Even when incorporating spike-history filters, the model with conditionally Poisson spiking (red) fails the time-rescaling test at the 95% level, though not so badly as the the inhomogeneous Poisson model (blue). However, the conditional Poisson model with spike-history filter (red) outperforms the non-parametric renewal model without spike-history filter (dark gray) on likelihood-based cross-validation, carrying 14% more predictive information. For this neuron, incorporating non-Poisson renewal properties into a model with spike history dependent intensity (light gray) provides only a modest (<1%) increase in cross-validation performance. Thus, in addition to being more tractable for estimation, it appears that the generalized linear modeling framework captures spike-train dependencies more accurately than a non-Poisson renewal process (at least for this neuron). We are in the process of applying this analysis to more data.

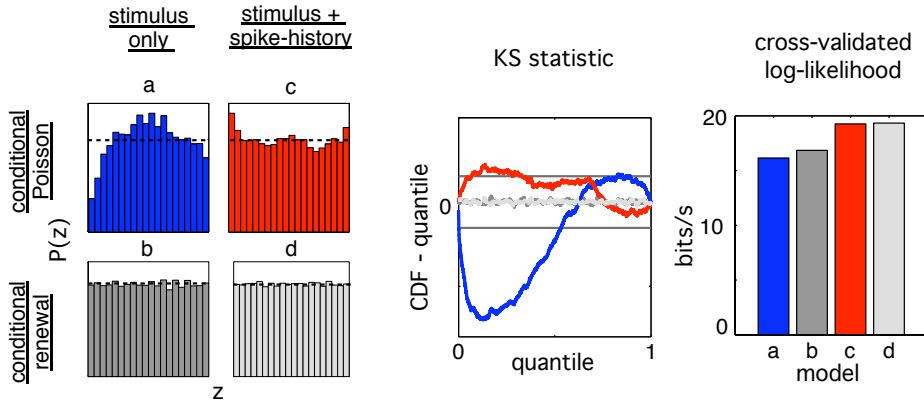

**Figure 4:** Evaluation of four specific cases of the conditional renewal model, fit to spike responses from a retinal ganglion cell stimulated with a time-varying white noise stimulus. **Left:** marginal distribution over the interspike intervals $\{z_i\}$, rescaled according to their cdf defined under four different models: (a) Inhomogeneous Poisson (i.e., LNP) model, without spike-history filter. (b) Conditional renewal model without spike-history filter, with non-parametrically estimated renewal density $\phi$. (c) Conditional Poison model, with spike-history filter (GLM). (d) Conditional renewal model with spike-history filter *and* non-parametrically estimated renewal density. A uniform distribution indicates good model fit under the time-rescaling test. **Middle:** The difference between the empirical cdf of the rescaled intervals (under all four models) and their quantiles. As expected, (a) fares poorly, (c) performs better but slightly exceeds the 95% confidence interval (black lines), and (b) and (d) exhibit near-perfect time-rescaling properties. **Right**: Likelihood-based cross-validation performance. Adding a non-parametric renewal density adds 4% to the Poisson model performance, but <1% to the GLM performance. Overall, a spike-history filter improves cross-validation performance more than the use of non-Poisson renewal process.

# 7   Discussion

We have connected two basic approaches for incorporating spike-history effects into neural encoding models: (1) non-Poisson renewal processes; and (2) conditionally Poisson processes with an intensity that depends on spike train history. We have shown that both kinds of effects can be regarded as special cases of a conditional renewal (CR) process model, and have formulated the model likelihood in a manner that separates the contributions from these two kinds of mechanisms.

Additionally, we have derived a condition on the CR model renewal density under which the likelihood function over filter parameters is log-concave, guaranteeing that ML estimation of filters (and MAP stimulus decoding) is a convex optimization problem.

We have shown that incorporating a non-parametric estimate of the CR model renewal density ensures near-perfect performance on the time-rescaling goodness-of-fit test, even when the model itself has little predictive accuracy (e.g., due to a poor model of the base intensity). Thus, we would argue that K-S tests based on the time-rescaled interspike intervals should not be used in isolation, but rather in conjunction with other tools for model comparison (e.g., cross-validated log-likelihood). Failure under the time-rescaling test indicates that model performance may be improved by incorporating a non-Poisson renewal density, which as we have shown, may be estimated directly from rescaled intervals.

Finally, we have applied the CR model to neural data, and shown that it can capture spike-history dependencies in both real and rescaled time. In future work, we will examine larger datasets and explore whether rescaled-time or real-time models provide more accurate descriptions of the dependencies in spike trains from a wider variety of neural datasets.

### Acknowledgments

Thanks to E. J. Chichilnisky, A. M. Litke, A. Sher and J. Shlens for retinal data, and to J. Shlens and L. Paninski for helpful discussions.

## Footnotes

[1]Note that $\Lambda_{t_*}(t)$ is invertible for all spike times $t_i$, since necessarily $t_i \in \{t; \lambda(t) > 0\}$.

[2]A note on terminology: we follow [13] in defining $\lambda(t)$ to be the instantaneous rate for an inhomogeneous renewal process, which is *not* identical to the hazard function $H(t) = P(t_i \in [t, t + \Delta]|t_i > t_{i-1})/\Delta$, also known as the *conditional intensity* [1]. We will use "base intensity" for $\lambda(t)$ to avoid this confusion.

[3]For simplicity, we have ignored the intervals $(0, t_0]$, the time to the first spike, and $(t_n, T]$, the time after the last spike, which are simple to compute but contribute only a small fraction to the total likelihood.

[4]Allowed nonlinearities must grow monotonically, at least linearly and at most exponentially: e.g., $\exp(x)$; $\log(1 + \exp(x))$; $\lfloor x \rfloor^p$, $p \geq 1$.

[5]To see this, note that if $g$ is concave ($g'' \leq 0$) and non-increasing ($g' \leq 0$), and $f$ is convex ($f'' \geq 0$), then $\frac{d^2}{dx^2} g(f(x)) = g''(f(x))f'(x)^2 + g'(f(x))f''(x) \leq 0$, implying $g(f(x))$ is concave.

[6]ML estimation of the spline parameters is a convex problem with one linear equality constraint $\int_0^1 \phi(v)dv = 1$ and a family of inequality constraints $q(v) \geq 0, \forall v$, which can be optimized efficiently.

[7]Although we have defined the time-rescaling transform using the base intensity instead of the conditional intensity as in [1], the resulting tests are equivalent provided the K-S test is applied using the appropriate distribution.

# References

[1] E. Brown, R. Barbieri, V. Ventura, R. Kass, and L. Frank. The time-rescaling theorem and its application to neural spike train data analysis. *Neural Computation*, 14:325–346, 2002.

[2] E. J. Chichilnisky. A simple white noise analysis of neuronal light responses. *Network: Computation in Neural Systems*, 12:199–213, 2001.

[3] E. P. Simoncelli, L. Paninski, J. W. Pillow, and O. Schwartz. Characterization of neural responses with stochastic stimuli. In M. Gazzaniga, editor, *The Cognitive Neurosciences, III*, chapter 23, pages 327–338. MIT Press, 2004.

[4] M. Berry and M. Meister. Refractoriness and neural precision. *Journal of Neuroscience*, 18:2200–2211, 1998.

[5] Daniel S. Reich, Ferenc Mechler, Keith P. Purpura, and Jonathan D. Victor. Interspike intervals, receptive fields, and information encoding in primary visual cortex. *J. Neurosci.*, 20(5):1964–1974, 2000.

[6] N. Brenner, W. Bialek, and R. de Ruyter van Steveninck. Adaptive rescaling optimizes information transmission. *Neuron*, 26:695–702, 2000.

[7] W. Gerstner. Population dynamics of spiking neurons: Fast transients, asynchronous states, and locking. *Neural Computation*, 12(1):43–89, 2000.

[8] P. Reinagel and R. C. Reid. Temporal coding of visual information in the thalamus. *Journal of Neuroscience*, 20:5392–5400, 2000.

[9] J. W. Pillow, L. Paninski, V. J. Uzzell, E. P. Simoncelli, and E. J. Chichilnisky. Prediction and decoding of retinal ganglion cell responses with a probabilistic spiking model. *The Journal of Neuroscience*, 25:11003–11013, 2005.

[10] M.A. Montemurro, S. Panzeri, M. Maravall, A. Alenda, M.R. Bale, M. Brambilla, and R.S. Petersen. Role of precise spike timing in coding of dynamic vibrissa stimuli in somatosensory thalamus. *Journal of Neurophysiology*, 98(4):1871, 2007.

[11] A.L. Jacobs, G. Fridman, R.M. Douglas, N.M. Alam, P. Latham, et al. Ruling out and ruling in neural codes. *Proceedings of the National Academy of Sciences*, 106(14):5936, 2009.

[12] M. Berman. Inhomogeneous and modulated gamma processes. *Biometrika*, 68(1):143–152, 1981.

[13] R. Barbieri, M.C. Quirk, L.M. Frank, M.A. Wilson, and E.N. Brown. Construction and analysis of non-poisson stimulus-response models of neural spiking activity. *Journal of Neuroscience Methods*, 105(1):25–37, 2001.

[14] E. Rossoni and J. Feng. A nonparametric approach to extract information from interspike interval data. *Journal of neuroscience methods*, 150(1):30–40, 2006.

[15] K. Koepsell and F.T. Sommer. Information transmission in oscillatory neural activity. *Biological Cybernetics*, 99(4):403–416, 2008.

[16] R.E. Kass and V. Ventura. A spike-train probability model. *Neural computation*, 13(8):1713–1720, 2001.

[17] W. Truccolo, U. T. Eden, M. R. Fellows, J. P. Donoghue, and E. N. Brown. A point process framework for relating neural spiking activity to spiking history, neural ensemble and extrinsic covariate effects. *J. Neurophysiol*, 93(2):1074–1089, 2005.

[18] J. W. Pillow, J. Shlens, L. Paninski, A. Sher, A. M. Litke, and E. P. Chichilnisky, E. J. Simoncelli. Spatio-temporal correlations and visual signaling in a complete neuronal population. *Nature*, 454:995–999, 2008.

[19] S. Gerwinn, J.H. Macke, M. Seeger, and M. Bethge. Bayesian inference for spiking neuron models with a sparsity prior. *Advances in Neural Information Processing Systems*, 2008.

[20] I.H. Stevenson, J.M. Rebesco, L.E. Miller, and K.P. K
"ording. Inferring functional connections between neurons. *Current Opinion in Neurobiology*, 18(6):582–588, 2008.

[21] L. Paninski. Maximum likelihood estimation of cascade point-process neural encoding models. *Network: Computation in Neural Systems*, 15:243–262, 2004.

[22] J. W. Pillow. Likelihood-based approaches to modeling the neural code. In K. Doya, S. Ishii, A. Pouget, and R. P. Rao, editors, *Bayesian Brain: Probabilistic Approaches to Neural Coding*, pages 53–70. MIT Press, 2007.

[23] T.P. Coleman and S. Sarma. Using convex optimization for nonparametric statistical analysis of point processes. In *IEEE International Symposium on Information Theory, 2007. ISIT 2007*, pages 1476–1480, 2007.

[24] L. Paninski, M. Fellows, S. Shoham, N. Hatsopoulos, and J. Donoghue. Superlinear population encoding of dynamic hand trajectory in primary motor cortex. *J. Neurosci.*, 24:8551–8561, 2004.

